# WINNER-TAKE-ALL

# NETWORKS OF $O(N)$ COMPLEXITY

J. Lazzaro, S. Ryckebusch, M.A. Mahowald, and C. A. Mead
California Institute of Technology
Pasadena, CA 91125

## ABSTRACT

We have designed, fabricated, and tested a series of compact CMOS integrated circuits that realize the winner-take-all function. These analog, continuous-time circuits use only O(n) of interconnect to perform this function. We have also modified the winner-take-all circuit, realizing a circuit that computes local nonlinear inhibition.

Two general types of inhibition mediate activity in neural systems: subtractive inhibition, which sets a zero level for the computation, and multiplicative (nonlinear) inhibition, which regulates the gain of the computation. We report a physical realization of general nonlinear inhibition in its extreme form, known as *winner-take-all*.

We have designed and fabricated a series of compact, completely functional CMOS integrated circuits that realize the winner-take-all function, using the full analog nature of the medium. This circuit has been used successfully as a component in several VLSI sensory systems that perform auditory localization (Lazzaro and Mead, in press) and visual stereopsis (Mahowald and Delbruck, 1988). Winner-take-all circuits with over 170 inputs function correctly in these sensory systems.

We have also modified this global winner-take-all circuit, realizing a circuit that computes local nonlinear inhibition. The circuit allows multiple winners in the network, and is well suited for use in systems that represent a feature space topographically and that process several features in parallel. We have designed, fabricated, and tested a CMOS integrated circuit that computes locally the winner-take-all function of spatially ordered input.

## THE WINNER-TAKE-ALL CIRCUIT

Figure 1 is a schematic diagram of the winner-take-all circuit. A single wire, associated with the potential $V_c$, computes the inhibition for the entire circuit; for an $n$ neuron circuit, this wire is $O(n)$ long. To compute the global inhibition, each neuron $k$ contributes a current onto this common wire, using transistor $T_{2_k}$. To apply this global inhibition locally, each neuron responds to the common wire voltage $V_c$, using transistor $T_{1_k}$. This computation is continuous in time; no clocks are used. The circuit exhibits no hysteresis, and operates with a time constant related to the size of the largest input. The output representation of the circuit is not binary; the winning output encodes the logarithm of its associated input.

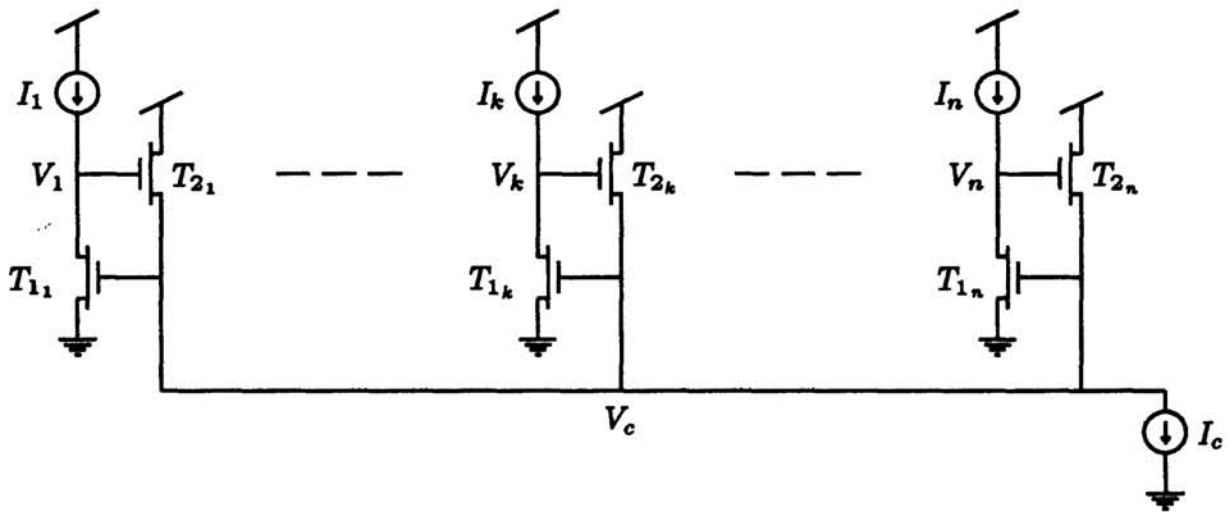

**Figure 1.** Schematic diagram of the winner-take-all circuit. Each neuron receives a unidirectional current input $I_k$; the output voltages $V_1 \ldots V_n$ represent the result of the winner-take-all computation. If $I_k = \max(I_1 \ldots I_n)$, then $V_k$ is a logarithmic function of $I_k$; if $I_j \ll I_k$, then $V_j \approx 0$.

A static and dynamic analysis of the two-neuron circuit illustrates these system properties. Figure 2 shows a schematic diagram of a two-neuron winner-take-all circuit. To understand the behavior of the circuit, we first consider the input condition $I_1 = I_2 \equiv I_m$. Transistors $T_{1_1}$ and $T_{1_2}$ have identical potentials at gate and source, and are both sinking $I_m$; thus, the drain potentials $V_1$ and $V_2$ must be equal. Transistors $T_{2_1}$ and $T_{2_2}$ have identical source, drain, and gate potentials, and therefore must sink the identical current $I_{c_1} = I_{c_2} = I_c/2$. In the subthreshold region of operation, the equation $I_m = I_o \exp(V_c/V_o)$ describes transistors $T_{1_1}$ and $T_{1_2}$, where $I_o$ is a fabrication parameter, and $V_o = kT/q\kappa$. Likewise, the equation $I_c/2 = I_o \exp((V_m - V_c)/V_o)$, where $V_m \equiv V_1 = V_2$, describes transistors $T_{2_1}$ and $T_{2_2}$. Solving for $V_m(I_m, I_c)$ yields

$$V_m = V_o \ln\left(\frac{I_m}{I_o}\right) + V_o \ln\left(\frac{I_c}{2I_o}\right). \tag{1}$$

Thus, for equal input currents, the circuit produces equal output voltages; this behavior is desirable for a winner-take-all circuit. In addition, the output voltage $V_m$ logarithmically encodes the magnitude of the input current $I_m$.

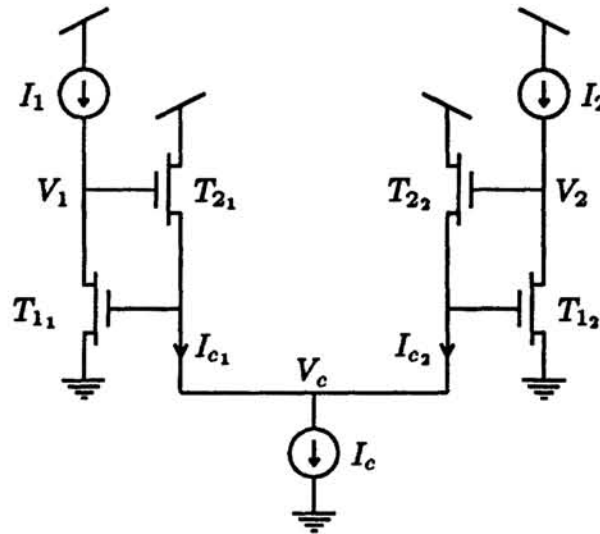

**Figure 2.** Schematic diagram of a two-neuron winner-take-all circuit.

The input condition $I_1 = I_m + \delta_i$, $I_2 = I_m$ illustrates the inhibitory action of the circuit. Transistor $T_{1_1}$ must sink $\delta_i$ more current than in the previous example; as a result, the gate voltage of $T_{1_1}$ rises. Transistors $T_{1_1}$ and $T_{1_2}$ share a common gate, however; thus, $T_{1_2}$ must also sink $I_m + \delta_i$. But only $I_m$ is present at the drain of $T_{1_2}$. To compensate, the drain voltage of $T_{1_2}$, $V_2$, must decrease. For small $\delta_i$s, the Early effect serves to decrease the current through $T_{1_2}$, decreasing $V_2$ linearly with $\delta_i$. For large $\delta_i$s, $T_{1_2}$ must leave saturation, driving $V_2$ to approximately 0 volts. As desired, the output associated with the smaller input diminishes. For large $\delta_i$s, $I_{c_2} \approx 0$, and $I_{c_1} \approx I_c$. The equation $I_m + \delta_i = I_o \exp(V_c/V_o)$ describes transistor $T_{1_1}$, and the equation $I_c = I_o \exp((V_1 - V_c)/V_o)$ describes transistor $T_{2_1}$. Solving for $V_1$ yields

$$V_1 = V_o \ln\left(\frac{I_m + \delta_i}{I_o}\right) + V_o \ln\left(\frac{I_c}{I_o}\right). \qquad (2)$$

The winning output encodes the logarithm of the associated input. The symmetrical circuit topology ensures similar behavior for increases in $I_2$ relative to $I_1$.

Equation 2 predicts the winning response of the circuit; a more complex expression, derived in (Lazzaro et.al., 1989), predicts the losing and crossover response of the circuit. Figure 3 is a plot of this analysis, fit to experimental data. Figure 4 shows the wide dynamic range and logarithmic properties of the circuit; the experiment in Figure 3 is repeated for several values of $I_2$, ranging over four orders of magnitude. The conductance of transistors $T_{1_1}$ and $T_{1_2}$ determines the losing response of the circuit. Variants of the winner-take-all circuit shown in (Lazzaro et. al., 1988) achieve losing responses wider and narrower than Figure 3, using circuit and mask layout techniques.

## WINNER-TAKE-ALL TIME RESPONSE

A good winner-take-all circuit should be stable, and should not exhibit damped oscillations ("ringing") in response to input changes. This section explores these dynamic properties of our winner-take-all circuit, and predicts the temporal response of the circuit. Figure 8 shows the two-neuron winner-take-all circuit, with capacitances added to model dynamic behavior.

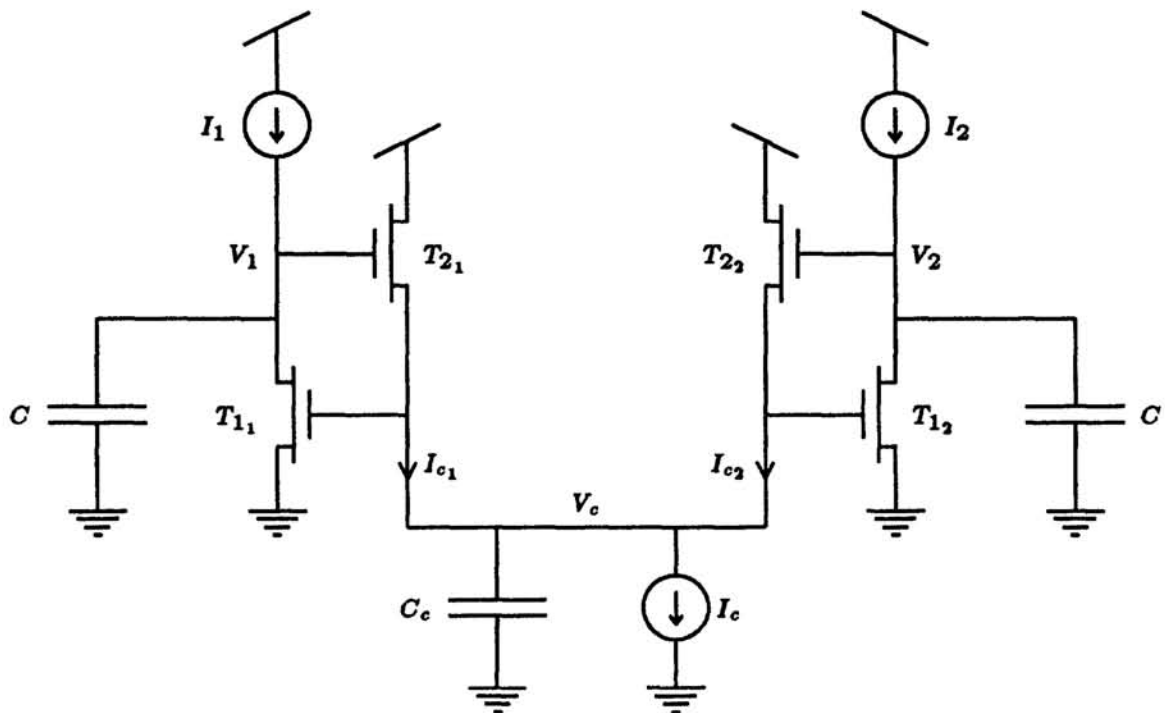

**Figure 8.** Schematic diagram of a two-neuron winner-take-all circuit, with capacitances added for dynamic analysis. $C$ is a large MOS capacitor added to each neuron for smoothing; $C_c$ models the parasitic capacitance contributed by the gates of $T_{11}$ and $T_{12}$, the drains of $T_{21}$ and $T_{22}$, and the interconnect.

(Lazzaro et. al., 1988) shows a small-signal analysis of this circuit. The transfer function for the circuit has real poles, and thus the circuit is stable and does not ring, if $I_c > 4I(C_c/C)$, where $I_1 \approx I_2 \approx I$. Figure 9 compares this bound with experimental data.

If $I_c > 4I(C_c/C)$, the circuit exhibits first-order behavior. The time constant $CV_o/I$ sets the dynamics of the winning neuron, where $V_o = kT/q\kappa \approx 40$ mV. The time constant $CV_E/I$ sets the dynamics of the losing neuron, where $V_E \approx 50\,V$. Figure 10 compares these predictions with experimental data.

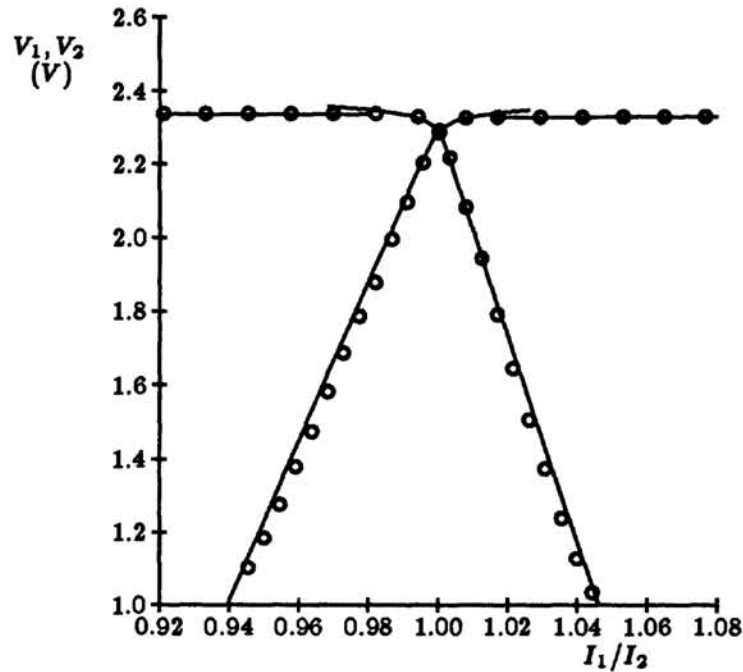

**Figure 3.** Experimental data (circles) and theory (solid lines) for a two-neuron winner-take-all circuit. $I_1$, the input current of the first neuron, is swept about the value of $I_2$, the input current of the second neuron; neuron voltage outputs $V_1$ and $V_2$ are plotted versus normalized input current.

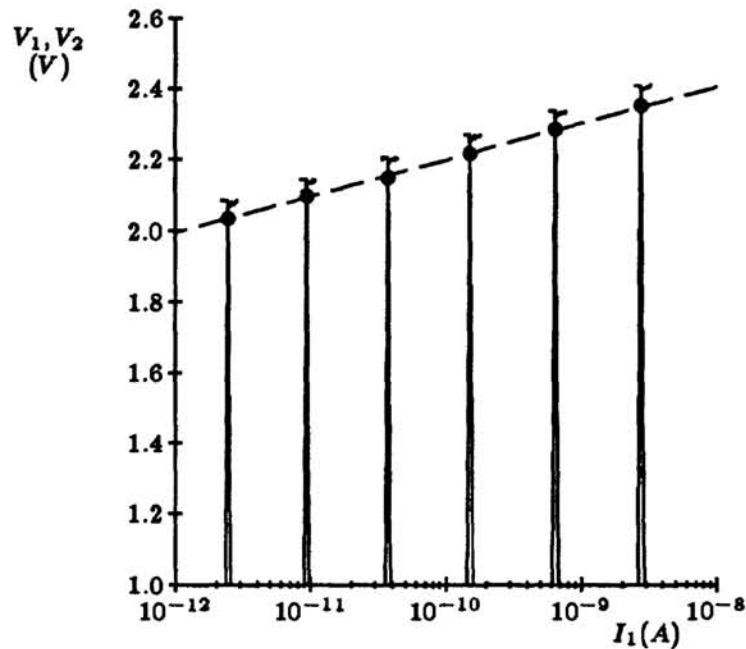

**Figure 4.** The experiment of Figure 3 is repeated for several values of $I_2$; experimental data of output voltage response are plotted versus absolute input current on a log scale. The output voltage $V_1 = V_2$ is highlighted with a circle for each experiment. The dashed line is a theoretical expression confirming logarithmic behavior over four orders of magnitude (Equation 1).

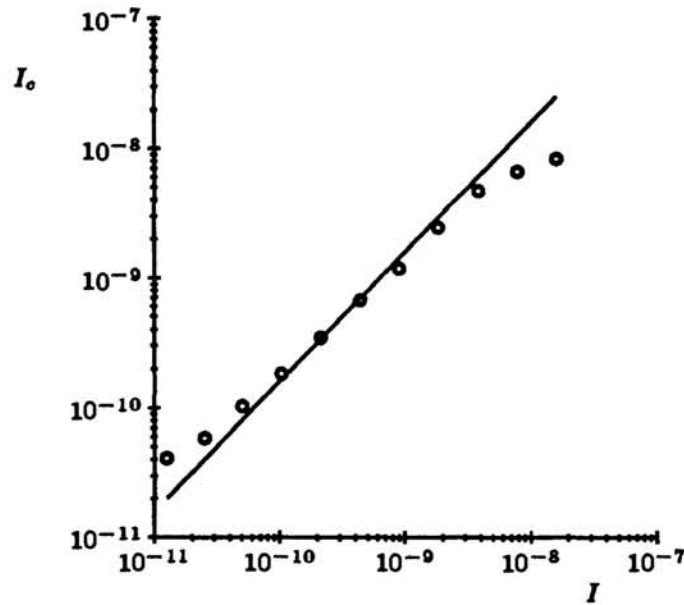

**Figure 9.** Experimental data (circles) and theoretical statements (solid line) for a two-neuron winner-take-all circuit, showing the smallest $I_c$, for a given $I$, necessary for a first-order response to a small-signal step input.

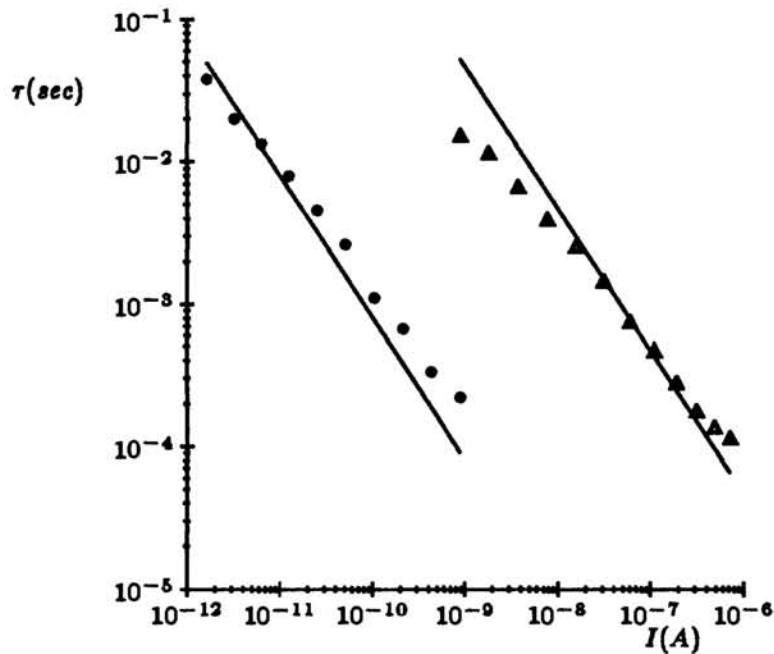

**Figure 10.** Experimental data (symbols) and theoretical statements (solid line) for a two-neuron winner-take-all circuit, showing the time constant of the first-order response to a small-signal step input. The winning response (filled circles) and losing response (triangles) of a winner-take-all circuit are shown; the time constants differ by several orders of magnitude.

## THE LOCAL NONLINEAR INHIBITION CIRCUIT

The winner-take-all circuit in Figure 1, as previously explained, locates the largest input to the circuit. Certain applications require a gentler form of nonlinear inhibition. Sometimes, a circuit that can represent multiple intensity scales is necessary. Without circuit modification, the winner-take-all circuit in Figure 1 can perform this task. (Lazzaro et. al., 1988) explains this mode of operation.

Other applications require a local winner-take-all computation, with each winner having influence over only a limited spatial area. Figure 12 shows a circuit that computes the local winner-take-all function. The circuit is identical to the original winner-take-all circuit, except that each neuron connects to its nearest neighbors with a nonlinear resistor circuit (Mead, in press). Each resistor conducts a current $I_r$ in response to a voltage $\Delta V$ across it, where $I_r = I_s \tanh(\Delta V/(2V_o))$. $I_s$, the saturating current of the resistor, is a controllable parameter. The current source, $I_o$, present in the original winner-take-all circuit, is distributed between the resistors in the local winner-take-all circuit.

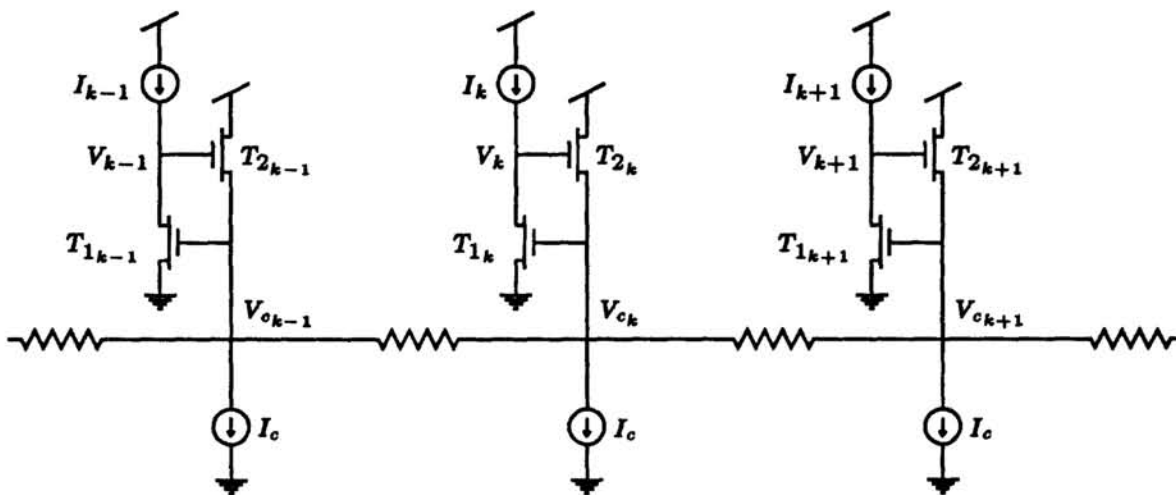

**Figure 11.** Schematic diagram of a section of the local winner-take-all circuit. Each neuron $i$ receives a unidirectional current input $I_i$; the output voltages $V_i$ represent the result of the local winner-take-all computation.

To understand the operation of the local winner-take-all circuit, we consider the circuit response to a spatial impulse, defined as $I_k \gg I$, where $I \equiv I_{i\neq k}$. $I_k \gg I_{k-1}$ and $I_k \gg I_{k+1}$, so $V_{c_k}$ is much larger than $V_{c_{k-1}}$ and $V_{c_{k+1}}$, and the resistor circuits connecting neuron $k$ with neuron $k-1$ and neuron $k+1$ saturate. Each resistor sinks $I_s$ current when saturated; transistor $T_{2_k}$ thus conducts $2I_s + I_c$ current. In the subthreshold region of operation, the equation $I_k = I_o \exp(V_{c_k}/V_o)$ describes transistor $T_{1_k}$, and the equation $2I_s + I_c = I_o \exp((V_k - V_{c_k})/V_o)$ describes transistor

$T_{2_k}$. Solving for $V_k$ yields

$$V_k = V_o \ln\big((2I_s + I_c)/I_o\big) + V_o \ln\big(I_k/I_o\big). \qquad (4)$$

As in the original winner-take-all circuit, the output of a winning neuron encodes the logarithm of that neuron's associated input.

As mentioned, the resistor circuit connecting neuron $k$ with neuron $k-1$ sinks $I_s$ current. The current sources $I_c$ associated with neurons $k-1$, $k-2$, ... must supply this current. If the current source $I_c$ for neuron $k-1$ supplies part of this current, the transistor $T_{2_{k-1}}$ carries no current, and the neuron output $V_{k-1}$ approaches zero. In this way, a winning neuron inhibits its neighboring neurons.

This inhibitory action does not extend throughout the network. Neuron $k$ needs only $I_s$ current from neurons $k-1$, $k-2$, .... . Thus, neurons sufficiently distant from neuron $k$ maintain the service of their current source $I_o$, and the outputs of these distant neurons can be active. Since, for a spatial impulse, all neurons $k-1$, $k-2$, ... have an equal input current $I$, all distant neurons have the equal output

$$V_{i \ll k} = V_o \ln\big(I_c/I_o\big) + V_o \ln\big(I/I_o\big). \qquad (5)$$

Similar reasoning applies for neurons $k+1$, $k+2$, .... .

The relative values of $I_s$ and $I_c$ determine the spatial extent of the inhibitory action. Figure 12 shows the spatial impulse response of the local winner-take-all circuit, for different settings of $I_s/I_c$.

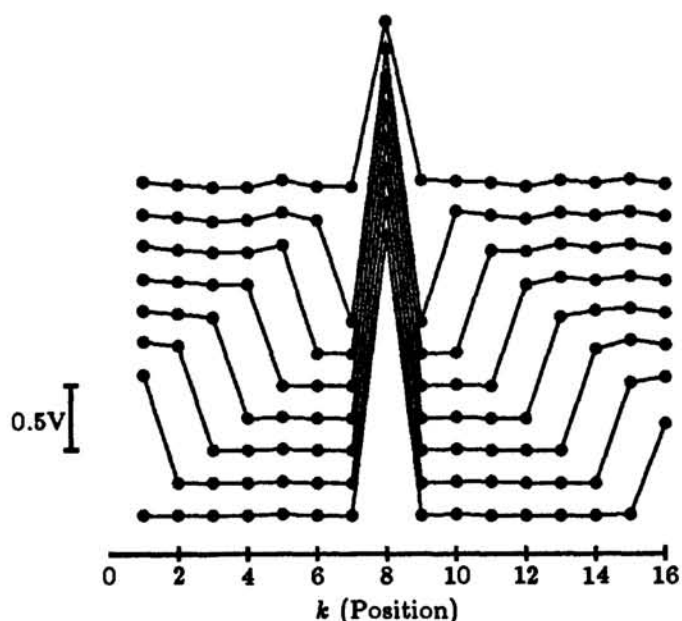

**Figure 12.** Experimental data showing the spatial impulse response of the local winner-take-all circuit, for values of $I_s/I_c$ ranging over a factor of 12.7. Wider inhibitory responses correspond to larger ratios. For clarity, the plots are vertically displaced in 0.25 volt increments.

# CONCLUSIONS

The circuits described in this paper use the full analog nature of MOS devices to realize an interesting class of neural computations efficiently. The circuits exploit the physics of the medium in many ways. The winner-take-all circuit uses a single wire to compute and communicate inhibition for the entire circuit. Transistor $T_{1_k}$ in the winner-take-all circuit uses two physical phenomena in its computation: its exponential current function encodes the logarithm of the input, and the finite conductance of the transistor defines the losing output response. As evolution exploits all the physical properties of neural devices to optimize system performance, designers of synthetic neural systems should strive to harness the full potential of the physics of their media.

## Acknowledgments

John Platt, John Wyatt, David Feinstein, Mark Bell, and Dave Gillespie provided mathematical insights in the analysis of the circuit. Lyn Dupré proofread the document. We thank Hewlett-Packard for computing support, and DARPA and MOSIS for chip fabrication. This work was sponsored by the Office of Naval Research and the System Development Foundation.

## References

Lazzaro, J. P., Ryckebusch, S., Mahowald, M.A., and Mead, C.A. (1989). *Winner-Take-All Networks of O(N) Complexity*, Caltech Computer Science Department Technical Report Caltech–CS–TR–21–88.

Lazzaro, J. P., and Mead, C.A. (in press). Silicon Models of Auditory Localization, *Neural Computation*.

Mahowald, M.A., and Delbruck, T.I. (1988). An Analog VLSI Implementation of the Marr–Poggio Stereo Correspondence Algorithm, *Abstracts of the First Annual INNS Meeting*, Boston, 1988, Vol. 1, Supplement 1, p. 392.

Mead, C. A. (in press). *Analog VLSI and Neural Systems*. Reading, MA: Addison-Wesley.
